# Multiple-Instance Pruning For Learning Efficient Cascade Detectors

**Cha Zhang and Paul Viola**
Microsoft Research
One Microsoft Way, Redmond, WA 98052
{chazhang,viola}@microsoft.com

## Abstract

Cascade detectors have been shown to operate extremely rapidly, with high accuracy, and have important applications such as face detection. Driven by this success, cascade learning has been an area of active research in recent years. Nevertheless, there are still challenging technical problems during the training process of cascade detectors. In particular, determining the optimal target detection rate for each stage of the cascade remains an unsolved issue. In this paper, we propose the multiple instance pruning (MIP) algorithm for soft cascades. This algorithm computes a set of thresholds which aggressively terminate computation with *no reduction in detection rate or increase in false positive rate* on the training dataset. The algorithm is based on two key insights: i) examples that are destined to be rejected by the complete classifier can be safely pruned early; ii) face detection is a multiple instance learning problem. The MIP process is fully automatic and requires no assumptions of probability distributions, statistical independence, or ad hoc intermediate rejection targets. Experimental results on the MIT+CMU dataset demonstrate significant performance advantages.

## 1  Introduction

The state of the art in real-time face detection has progressed rapidly in recently years. One very successful approach was initiated by Viola and Jones [11]. While some components of their work are quite simple, such as the so called "integral image", or the use of AdaBoost, a great deal of complexity lies in the training of the cascaded detector. There are many required parameters: the number and shapes of rectangle filters, the number of stages, the number of weak classifiers in each stage, and the target detection rate for each cascade stage. These parameters conspire to determine not only the ROC curve for the resulting system but also its computational complexity. Since the Viola-Jones training process requires CPU days to train and evaluate, it is difficult, if not impossible, to pick these parameters optimally.

The conceptual and computational complexity of the training process has led to many papers proposing improvements and refinements [1, 2, 4, 5, 9, 14, 15]. Among them, three are closely related to this paper: Xiao, Zhu and Zhang[15], Sochman and Matas[9], and Bourdev and Brandt[1]. In each paper, the original cascade structure of distinct and separate stages is relaxed so that earlier computation of weak classifier scores can be combined with later weak classifiers. Bourdev and Brandt coined the term, "soft-cascade", where the entire detector is trained as a single strong classifier without stages (with 100's or 1000's of weak classifiers sometimes called "features"). The score assigned to a detection window by the soft cascade is simply a weighted sum of the weak classifiers: $s_k(T) = \sum_{j \leq T} \alpha_j h_j(x_k)$, where $T$ is the total number of weak classifiers; $h_j(x_k)$ is the $j^{th}$ feature computed on example $x_k$; $\alpha_j$ is the vote on weak classifier $j$. Computation of the sum is terminated early whenever the partial sum falls below a rejection threshold: $s_k(t) < \theta(t)$. Note the soft cascade

is similar to, but simpler than both the boosting chain approach of Xiao, Zhu, and Zhang and the WaldBoost approach of Sochman and Matas.

The rejection thresholds $\theta(t), t \in \{1, \cdots, T - 1\}$ are critical to the performance and speed of the complete classifier. However, it is difficult to set them optimally in practice. One possibility is to set the rejection thresholds so that no positive example is lost; this leads to very conservative thresholds and a very slow detector. Since the complete classifier will not achieve 100% detection (Note, given practical considerations, the final threshold of the complete classifier is set to reject some positive examples because they are difficult to detect. Reducing the final threshold further would admit too many false positives.), it seems justified to reject positive examples early in return for fast detection speed. The main question is which positive examples can be rejected and when.

A key criticism of all previous cascade learning approaches is that none has a scheme to determine which examples are best to reject. Viola-Jones attempted to reject zero positive examples until this become impossible and then reluctantly gave up on one positive example at a time. Bourdev and Brandt proposed a method for setting rejection thresholds based on an ad hoc detection rate target called a "rejection distribution vector", which is a parameterized exponential curve. Like the original Viola-Jones proposal, the soft-cascade gradually gives up on a number of positive examples in an effort to aggressively reduce the number of negatives passing through the cascade. Perhaps a particular family of curves is more palatable, but it is still arbitrary and non-optimal. Sochman-Matas used a ratio test to determine the rejection thresholds. While this has statistical validity, distributions must be estimated, which introduces empirical risk. This is a particular problem for the first few rejection thresholds, and can lead to low detection rates on test data.

This paper proposes a new mechanism for setting the rejection thresholds of any soft-cascade which is conceptually simple, has no tunable parameters beyond the final detection rate target, yet yields a cascade which is both highly accurate and very fast. Training data is used to set all reject thresholds after the final classifier is learned. There are no assumptions about probability distributions, statistical independence, or ad hoc intermediate targets for detection rate (or false positive rate). The approach is based on two key insights that constitute the major contributions of this paper: 1) positive examples that are rejected by the complete classifier can be safely rejected earlier during pruning; 2) each ground-truth face requires no more than one matched detection window to maintain the classifier's detection rate. We propose a novel algorithm, *multiple instance pruning* (MIP), to set the reject thresholds automatically, which results in a very efficient cascade detector with superior performance.

The rest of the paper is organized as follows. Section 2 describes an algorithm which makes use of the final classification results to perform pruning. Multiple instance pruning is presented in Section 3. Experimental results and conclusions are given in Section 4 and 5, respectively.

## 2   Pruning Using the Final Classification

We propose a scheme which is simultaneously simpler and more effective than earlier techniques. Our key insight is quite simple: the reject thresholds are set so that they give up on precisely those positive examples which are rejected by the complete classifier. Note that the score of each example, $s_k(t)$ can be considered a trajectory through time. The full classifier rejects a positive example if its final score $s_k(T)$ falls below the final threshold $\theta(T)$. In the simplest version of our threshold setting algorithm, all trajectories from positive windows which fall below the final threshold are removed. Each rejection threshold is then simply:

$$\theta(t) = \min_{\left\{k \middle| s_k(T) > \theta(T), y_k = 1\right\}} s_k(t)$$

where $\{x_k, y_k\}$ is the training set in which $y_k = 1$ indicates positive windows and $y_k = -1$ indicates negative windows. These thresholds produce a reasonably fast classifier which is guaranteed to produce no more errors than the complete classifier (on the training dataset). We call this pruning algorithm *direct backward pruning* (DBP).

One might question whether the minimum of all retained trajectories is robust to mislabeled or noisy examples in the training set. Note that the final threshold of the complete classifier will often reject mislabeled or noisy examples (though they will be considered false negatives). These rejected

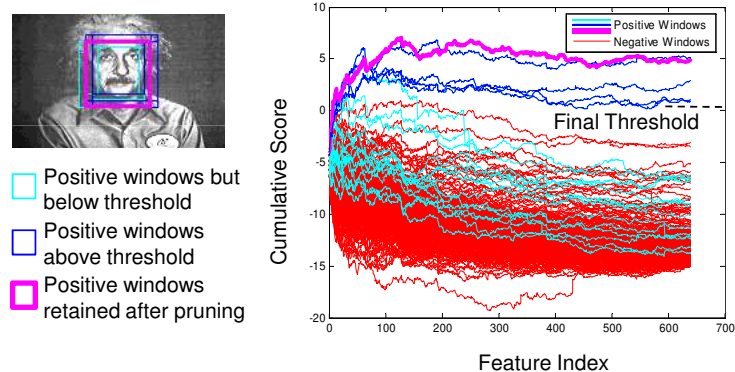

Figure 1: Traces of cumulative scores of different windows in an image of a face. See text.

examples play no role in setting the rejection thresholds. We have found this procedure very robust to the types of noise present in real training sets.

In past approaches, thresholds are set to reject the largest number of negative examples and only a small *percentage of positive examples*. These approaches justify these thresholds in different ways, but they all struggle to determine the correct percentage accurately and effectively. In the new approach, the final threshold of the *complete soft-cascade* is set to achieve the require detection rate. Rejection thresholds are then set to reject the largest number of negative examples and retain *all* positive examples which are retained by the complete classifier. The important difference is that the particular positive examples which are rejected are those which are destined to be rejected by the final classifier. This yields a fast classifier which *labels all positive examples in exactly the same way as the complete classifier*. In fact, it yields the fastest possible soft-cascade with such property (provided the weak classifiers are not re-ordered). Note, some negative examples that eventually pass the complete classifier threshold may be pruned by earlier rejection thresholds. This has the satisfactory side benefit of reducing false positive rate as well. In contrast, although the detection rate on the training set can also be guaranteed in Bourdev-Brandt's algorithm, there is no guarantee that false positive rate will not increase.

Bourdev-Brandt propose reordering the weak classifiers based on the separation between the mean score of the positive examples and the mean score of the negative examples. Our approach is equally applicable to a reordered soft-cascade.

Figure 1 shows 293 trajectories from a single image whose final score is above -15. While the rejection thresholds are learned using a large set of training examples, this one image demonstrates the basic concepts. The red trajectories are negative windows. The single physical face is consistent with a set of positive detection windows that are within an acceptable range of positions and scales. Typically there are tens of acceptable windows for each face. The blue and magenta trajectories correspond to acceptable windows which fall above the final detection threshold. The cyan trajectories are potentially positive windows which fall below the final threshold. Since the cyan trajectories are rejected by the final classifier, rejection thresholds need only retain the blue and magenta trajectories.

In a sense the complete classifier, along with a threshold which sets the operating point, provides labels on examples which are *more valuable* than the ground-truth labels. There will always be a set of "positive" examples which are extremely difficult to detect, or worse which are mistakenly labeled positive. In practice the final threshold of the complete classifier will be set so that these *particular examples* are rejected. In our new approach these particular examples can be rejected early in the computation of the cascade. Compared with existing approaches, that set the reject thresholds in a heuristic manner, our approach is data-driven and hence more principled.

## 3 Multiple Instance Pruning

The notion of an "acceptable detection window" plays a critical role in an improved process for setting rejection thresholds. It is difficult to define the correct position and scale of a face in an image.

For a purely upright and frontal face, one might propose the smallest rectangle which includes the chin, forehead, and the inner edges of the ears. But, as we include a range of non-upright and non-frontal faces these rectangles can vary quite a bit. Should the correct window be a function of apparent head size? Or is eye position and interocular distance more reliable? Even given clear instructions, one finds that two subjects will differ significantly in their "ground-truth" labels.

Recall that the detection process scans the image generating a large, but finite, collection of over-lapping windows at various scales and locations. Even in the absence of ambiguity, some slop is required to ensure that at least one of the generated windows is considered a successful detection for each face. Experiments typically declare that any window which is within 50% in size and within a distance of 50% (of size) be considered a true positive. Using typical scanning parameters this can lead to tens of windows which are all equally valid positive detections. If any of these windows is classified positive then this face is consider detected.

Even though all face detection algorithms must address the "multiple window" issue, few papers have discussed it. Two papers which have fundamentally integrated this observation into the train-ing process are Nowlan and Platt [6] and more recently by Viola, Platt, and Zhang [12]. These papers proposed a multiple instance learning (MIL) framework where the positive examples are collected into "bags". The learning algorithm is then given the freedom to select at least one, and perhaps more examples, in each bag as the true positive examples. In this paper, we do not directly address soft-cascade learning, though we will incorporate the "multiple window" observation into the determination of the rejection thresholds.

One need only retain one "acceptable" window for each face which is detected by the final classifier. A more aggressive threshold is defined as:

$$\theta(t) = \min_{i \in P} \left[ \max_{\left\{ k \middle| k \in F_i \cap R_i, y_k = 1 \right\}} s_k(t) \right]$$

where $i$ is the index of ground-truth faces; $F_i$ is the set of acceptable windows associated with ground-truth face $i$ and $R_i$ is the set of windows which are "retained" (see below). $P$ is the set of ground-truth faces that have at least one acceptable window above the final threshold:

$$P = \left\{ i \middle| \max_{\left\{ k \middle| k \in F_i \right\}} s_k(T) > \theta(T) \right\}$$

In this new procedure the acceptable windows come in bags, only one of which must be classified positive in order to ensure that each face is successfully detected. This new criteria for success is more flexible and therefore more aggressive. We call this pruning method *multiple instance pruning* (MIP).

Returning to Figure 1 we can see that the blue, cyan, and magenta trajectories actually form a "bag". Both in this algorithm, and in the simpler previous algorithm, the cyan trajectories are rejected before the computation of the thresholds. The benefit of this new algorithm is that the blue trajectories can be rejected as well.

The definition of "retained" examples in the computation above is a bit more complex than before. Initially the trajectories from the positive bags which fall above the final threshold are retained. The set of retained examples is further reduced as the earlier thresholds are set. This is in contrast to the simpler DBP algorithm where the thresholds are set to preserve *all* retained positive examples. In the new algorithm the partial score of an example can fall below the current threshold (because it is in a bag with a better example). Each such example is removed from the retained set $R_i$ and not used to set subsequent thresholds.

The pseudo code of the MIP algorithm is shown in Figure 2. It guarantees the same face detection rate on the training dataset as the complete classifier. Note that the algorithm is greedy, setting earlier thresholds first so that all positive bags are retained and the fewest number of negative examples pass. Theoretically it is possible that delaying the rejection of a particular example may result in a better threshold at a later stage. Searching for the optimal MIP pruned detector, however, may be quite expensive. The MIP algorithm is however *guaranteed to generate a soft-cascade that is at least as fast as DBP*, since the criteria for setting the thresholds is less restrictive.

Figure 2: The MIP algorithm.

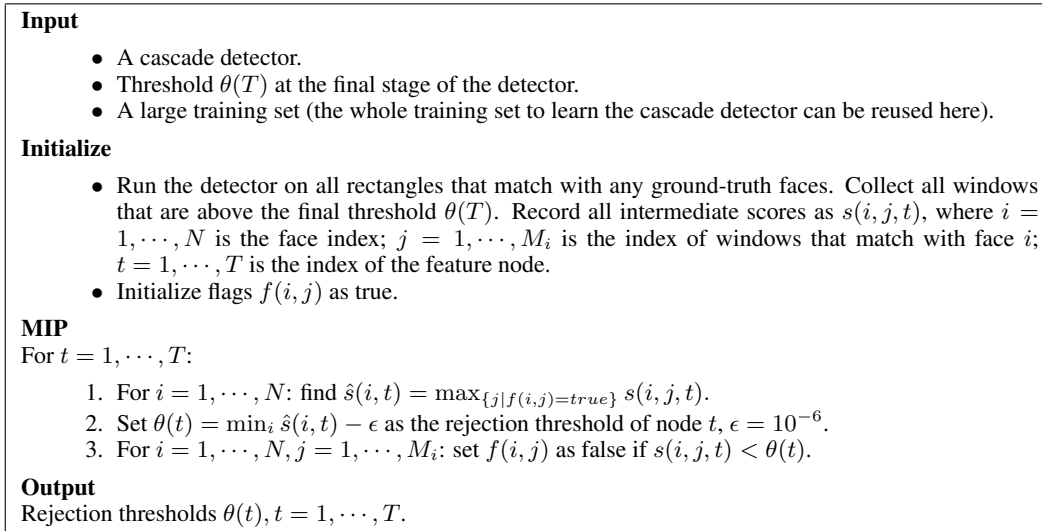

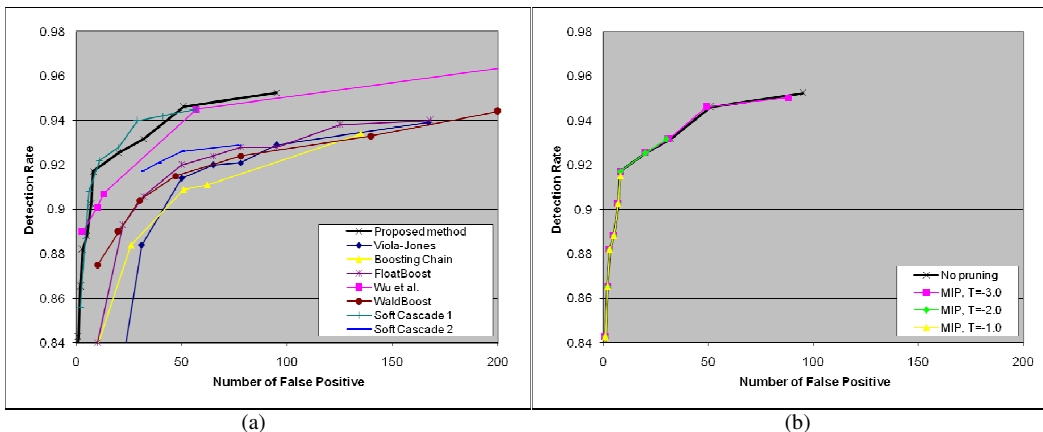

Figure 3: (a) Performance comparison with existing works on MIT+CMU frontal face dataset. (b) ROC curves of the detector after MIP pruning using the original training set. No performance degradation is found on the MIT+CMU testing dataset.

## 4 Experimental Results

More than 20,000 images were collected from the web, containing roughly 10,000 faces. Over 2 billion negative examples are generated from the same image set. A soft cascade classifier is learned through a new framework based on weight trimming and bootstrapping (see Appendix). The training process was conducted on a dual core AMD Opteron 2.2 GHz processor with 16 GB of RAM. It takes less than 2 days to train a classifier with 700 weak classifiers based on the Haar features [11]. The testing set is the standard MIT+CMU frontal face database [10, 7], which consists of 125 grayscale images containing 483 labeled frontal faces. A detected rectangle is considered to be a true detection if it has less than 50% variation in shift and scale from the ground-truth.

It is difficult to compare the performance of various detectors, since every detector is trained on a different dataset. Nevertheless, we show the ROC curves of a number of existing detectors and ours in Figure 3(a). Note there are two curves plotted for soft cascade. The first curve has very good performance, at the cost of slow speed (average 37.1 features per window). The classification accuracy dropped significantly in the second curve, which is faster (average 25 features per window).

| Final Threshold | -3.0 | -2.5 | -2.0 | -1.5 | -1.0 | -0.5 | 0.0 |
|---|---|---|---|---|---|---|---|
| Detection Rate | 95.2% | 94.6% | 93.2% | 92.5% | 91.7% | 90.3% | 88.8% |
| # of False Positive | 95 | 51 | 32 | 20 | 8 | 7 | 5 |
| DBP | 36.13 | 35.78 | 35.76 | 34.93 | 29.22 | 28.91 | 26.72 |
| MIP | 16.11 | 16.06 | 16.80 | 18.60 | 16.96 | 15.53 | 14.59 |

(a)

| Approach | Viola-Jones | Boosting chain | FloatBoost | WaldBoost | Wu et al. | Soft cascade |
|---|---|---|---|---|---|---|
| Total # of features | 6061 | 700 | 2546 | 600 | 756 | 4943 |
| Slowness | 10 | 18.1 | 18.9 | 13.9 | N/A | 37.1 (25) |

(b)

Figure 4: (a) Pruning performance of DBP and MIP. The bottom two rows indicate the average number of features visited per window on the MIT+CMU dataset. (b) Results of existing work.

Figure 4(a) compares DBP and MIP with different final thresholds of the strong classifier. The original data set for learning the soft cascade is reused for pruning the detector. Since MIP is a more aggressive pruning method, the average number of features evaluated is much lower than DBP. Note both DBP and MIP guarantee that no positive example from the *training set* is lost. There is no similar guarantee for test data, though. Figure 3(b) shows that there is no practical loss in classification accuracy on the MIT+CMU test dataset for various applications of the MIP algorithm (note that the MIT+CMU data is not used by the training process in any way).

Speed comparison with other algorithms are subtle (Figure 4(b)). The first observation is that higher detection rates almost always require the evaluation of additional features. This is certainly true in our experiments, but it is also true in past papers (e.g., the two curves of Bourdev-Brandt soft cascade in Figure 3(a)). The fastest algorithms often cannot achieve very high detection rates. One explanation is that in order to achieve higher detection rates one must retain windows which are "ambiguous" and may contain faces. The proposed MIP-based detector yields a much lower false positive rate than the 25-feature Bourdev-Brandt soft cascade and nearly 35% improvement on detection speed. While the WaldBoost algorithm is quite fast, detection rates are measurably lower. Detectors such as Viola-Jones, boosting chain, FloatBoost, and Wu et al. all requires manual tuning. We can only guess how much trial and error went into getting a fast detector that yields good results.

The expected computation time of the DBP soft-cascade varies monotonically in detection rate. This is guaranteed by the algorithm. In experiments with MIP we found a surprising quirk in the expected computation times. One would expect that if the required detection rate is higher, it world be more difficult to prune. In MIP, when the detection rate increases, there are two conflicting factors involved. First, the number of detected faces increases, which increases the difficulty of pruning. Second, for each face the number of retained and acceptable windows increases. Since we are computing the maximum of this larger set, MIP can in some cases be more aggressive. The second factor explains the increase of speed when the final threshold changes from -1.5 to -2.0.

The direct performance comparison between MIP and Bourdev-Brandt (B-B) was performed using the same soft-cascade and the same data. In order to better measure performance differences we created a larger test set containing 3,859 images with 3,652 faces collected from the web. Both algorithms prune the strong classifier for a target detection rate of 97.2% on the *training set*, which corresponds to having a final threshold of $-2.5$ in Figure 4(a). We use the same exponential function family as [1] for B-B, and adjust the control parameter $\alpha$ in the range between $-16$ and $4$. The results are shown in Figure 5. It can be seen that the MIP pruned detector has the best detection performance. When a positive $\alpha$ is used (e.g., $\alpha = 4$), the B-B pruned detector is still worse than the MIP pruned detector, and its speed is 5 times slower (56.83 vs. 11.25). On the other hand, when $\alpha$ is negative, the speed of B-B pruned detectors improves and can be faster than MIP (e.g., when $\alpha = -16$). Note, adjusting $\alpha$ leads to changes both in detection time and false positive rate.

In practice, both MIP and B-B can be useful. MIP is fully automated and guarantees detection rate with no increase in false positive rate on the training set. The MIP pruned strong classifier is usually fast enough for most real-world applications. On the other hand, if speed is the dominant factor, one can specify a target detection rate and target execution time and use B-B to find a solution.

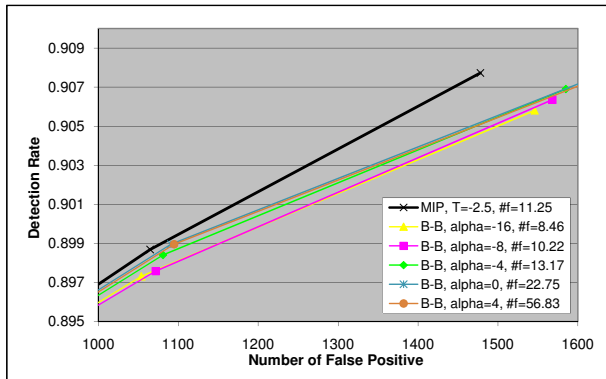

Figure 5: The detector performance comparison after applying MIP and Bourdev-Brandt's method [1]. Note, this test was done using a much larger, and more difficult, test set than MIT+CMU. In the legend, symbol $\#f$ represents the average number of weak classifiers visited per window.

Note such a solution is not guaranteed, and the false positive rate may be unacceptably high (The performance degradation of B-B heavily depends on the given soft-cascade. While with our detector the performance of B-B is acceptable even when $\alpha = -16$, the performance of the detector in [1] drops significantly from 37 features to 25 features, as shown in Fig. 3 (a).).

## 5 Conclusions

We have presented a simple yet effective way to set the rejection thresholds of a given soft-cascade, called multiple instance pruning (MIP). The algorithm begins with a conventional strong classifier and an associated final threshold. MIP then adds a set of rejection thresholds to construct a cascade detector. The rejection thresholds are determined so that every face which was detected by the original strong classifier is guaranteed to be detected by the soft cascade. The algorithm also guarantees that the false positive rate on the training set will not increase. There is only one parameter used throughout the cascade training process, the target detection rate for the final system. Moreover, there are no required assumptions about probability distributions, statistical independence, or ad hoc intermediate targets for detection rate or false positive rate.

## Appendix: Learn Soft Cascade with Weight Trimming and Bootstrapping

We present an algorithm for learning a strong classifier from a very large set of training examples. In order to deal with the many millions of examples, the learning algorithm uses both weight trimming and bootstrapping. Weight trimming was proposed by Friedman, Hastie and Tibshirani [3]. At each round of boosting it ignores training examples with the smallest weights, up to a percentage of the total weight which can be between 1% and 10%. Since the weights are typically very skewed toward a small number of hard examples, this can eliminate a very large number of examples. It was shown that weight trimming can dramatically reduce computation for boosted methods without sacrificing accuracy. In weight trimming no example is discarded permanently, therefore it is ideal for learning a soft cascade.

The algorithm is described in Figure 6. In step 4, a set $A$ is predefined to reduce the number of weight updates on the whole training set. One can in theory update the scores of the whole training set after each feature is learned if computationally affordable, though the gain in detector performance may not be visible.Note, a set of thresholds are also returned by this process (making the result a soft-cascade). These preliminary rejection thresholds are extremely conservative, retaining *all* positive examples in the training set. They result in a very slow detector – the average number of features visited per window is on the order of hundreds. These thresholds will be replaced with the ones derived by the MIP algorithm. We set the preliminary thresholds only to moderately speed up the computation of ROC curves before MIP.

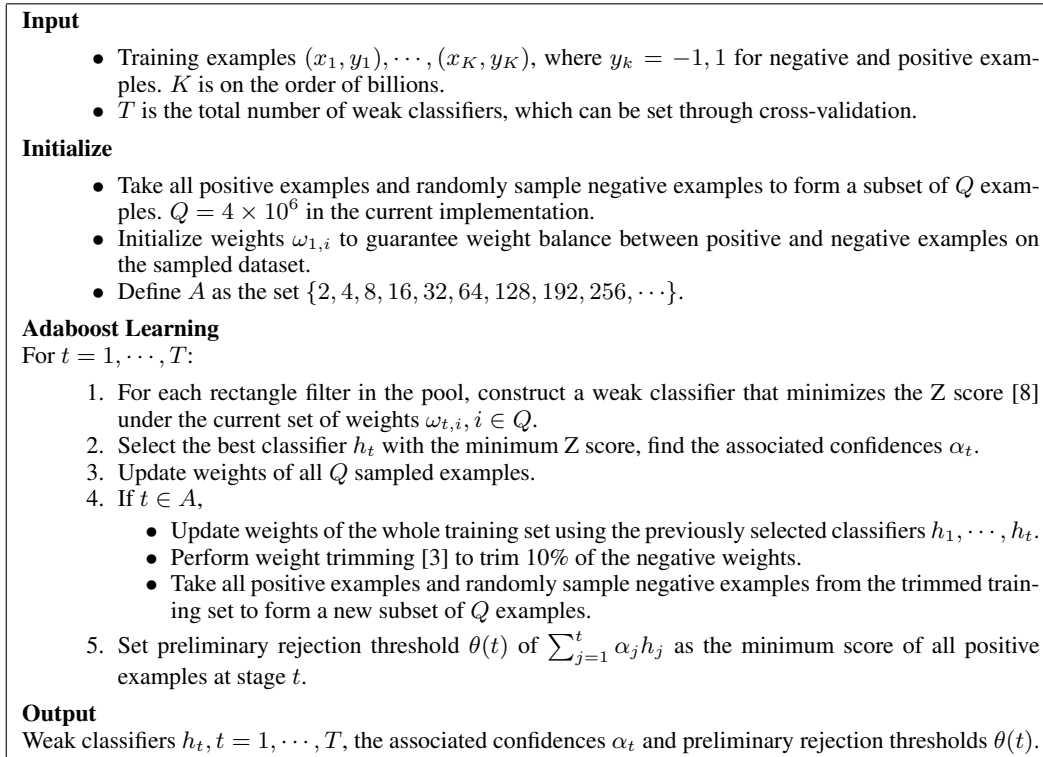

**Input**

- Training examples $(x_1, y_1), \cdots, (x_K, y_K)$, where $y_k = -1, 1$ for negative and positive examples. $K$ is on the order of billions.
- $T$ is the total number of weak classifiers, which can be set through cross-validation.

**Initialize**

- Take all positive examples and randomly sample negative examples to form a subset of $Q$ examples. $Q = 4 \times 10^6$ in the current implementation.
- Initialize weights $\omega_{1,i}$ to guarantee weight balance between positive and negative examples on the sampled dataset.
- Define $A$ as the set $\{2, 4, 8, 16, 32, 64, 128, 192, 256, \cdots\}$.

**Adaboost Learning**

For $t = 1, \cdots, T$:

1. For each rectangle filter in the pool, construct a weak classifier that minimizes the Z score [8] under the current set of weights $\omega_{t,i}, i \in Q$.
2. Select the best classifier $h_t$ with the minimum Z score, find the associated confidences $\alpha_t$.
3. Update weights of all $Q$ sampled examples.
4. If $t \in A$,
   - Update weights of the whole training set using the previously selected classifiers $h_1, \cdots, h_t$.
   - Perform weight trimming [3] to trim 10% of the negative weights.
   - Take all positive examples and randomly sample negative examples from the trimmed training set to form a new subset of $Q$ examples.
5. Set preliminary rejection threshold $\theta(t)$ of $\sum_{j=1}^{t} \alpha_j h_j$ as the minimum score of all positive examples at stage $t$.

**Output**

Weak classifiers $h_t, t = 1, \cdots, T$, the associated confidences $\alpha_t$ and preliminary rejection thresholds $\theta(t)$.

Figure 6: Adaboost learning with weight trimming and booststrapping.

## References

[1] L. Bourdev and J. Brandt. Robust object detection via soft cascade. In *Proc. of CVPR*, 2005.

[2] S. C. Brubaker, M. D. Mullin, and J. M. Rehg. Towards optimal training of cascaded detectors. In *Proc. of ECCV*, 2006.

[3] J. Friedman, T. Hastie, and R. Tibshirani. Additive logistic regression: a statistical view of boosting. Technical report, Dept. of Statistics, Stanford University, 1998.

[4] S. Li, L. Zhu, Z. Zhang, A. Blake, H. Zhang, and H. Shum. Statistical learning of multi-view face detection. In *Proc. of ECCV*, 2002.

[5] H. Luo. Optimization design of cascaded classifiers. In *Proc. of CVPR*, 2005.

[6] S. J. Nowlan and J. C. Platt. A convolutional neural network hand tracker. In *Proc. of NIPS*, volume 7, 1995.

[7] H. Rowley, S. Baluja, and T. Kanade. Neural network-based face detection. *IEEE Trans. on PAMI*, 20:23–38, 1998.

[8] R. E. Schapire and Y. Singer. Improved boosting algorithms using confidence-rated predictions. *Machine Learning*, 37:297–336, 1999.

[9] J. Sochman and J. Matas. Waldboost - learning for time constrained sequential detection. In *Proc. of CVPR*, 2005.

[10] K. Sung and T. Poggio. Example-based learning for view-based face detection. *IEEE Trans. on PAMI*, 20:39–51, 1998.

[11] P. Viola and M. Jones. Rapid object detection using a boosted cascade of simple features. In *Proc. of CVPR*, 2001.

[12] P. Viola, J. C. Platt, and C. Zhang. Multiple instance boosting for object detection. In *Proc. of NIPS*, volume 18, 2006.

[13] B. Wu, H. Ai, C. Huang, and S. Lao. Fast rotation invariant multi-view face detection based on real adaboost. In *Proc. of IEEE Automatic Face and Gesture Recognition*, 2004.

[14] J. Wu, J. M. Rehg, and M. D. Mullin. Learning a rare event detection cascade by direct feature selection. In *Proc. of NIPS*, volume 16, 2004.

[15] R. Xiao, L. Zhu, and H. Zhang. Boosting chain learning for object detection. In *Proc. of ICCV*, 2003.

